# An Application of Reinforcement Learning to Aerobatic Helicopter Flight

**Pieter Abbeel, Adam Coates, Morgan Quigley, Andrew Y. Ng**
Computer Science Dept.
Stanford University
Stanford, CA 94305

## Abstract

Autonomous helicopter flight is widely regarded to be a highly challenging control problem. This paper presents the first successful autonomous completion on a real RC helicopter of the following four aerobatic maneuvers: forward flip and sideways roll at low speed, tail-in funnel, and nose-in funnel. Our experimental results significantly extend the state of the art in autonomous helicopter flight. We used the following approach: First we had a pilot fly the helicopter to help us find a helicopter dynamics model and a reward (cost) function. Then we used a reinforcement learning (optimal control) algorithm to find a controller that is optimized for the resulting model and reward function. More specifically, we used differential dynamic programming (DDP), an extension of the linear quadratic regulator (LQR).

## 1 Introduction

Autonomous helicopter flight represents a challenging control problem with high-dimensional, asymmetric, noisy, nonlinear, non-minimum phase dynamics. Helicopters are widely regarded to be significantly harder to control than fixed-wing aircraft. (See, e.g., [14, 20].) At the same time, helicopters provide unique capabilities, such as in-place hover and low-speed flight, important for many applications. The control of autonomous helicopters thus provides a challenging and important testbed for learning and control algorithms.

In the "upright flight regime" there has recently been considerable progress in autonomous helicopter flight. For example, Bagnell and Schneider [6] achieved sustained autonomous hover. Both LaCivita et al. [13] and Ng et al. [17] achieved sustained autonomous hover and accurate flight in regimes where the helicopter's orientation is fairly close to upright. Roberts et al. [18] and Saripalli et al. [19] achieved vision based autonomous hover and landing. In contrast, autonomous flight achievements in other flight regimes have been very limited. Gavrilets et al. [9] achieved a split-S, a stall turn and a roll in forward flight. Ng et al. [16] achieved sustained autonomous inverted hover.

The results presented in this paper significantly expand the limited set of successfully completed aerobatic maneuvers. In particular, we present the first successful autonomous completion of the following four maneuvers: forward flip and axial roll at low speed, tail-in funnel, and nose-in funnel. Not only are we first to autonomously complete such a single flip and roll, our controllers are also able to continuously repeat the flips and rolls without any pauses in between. Thus the controller has to provide continuous feedback *during* the maneuvers, and cannot, for example, use a period of hovering to correct errors of the first flip before performing the next flip. The number of flips and rolls and the duration of the funnel trajectories were chosen to be sufficiently large to demonstrate that the helicopter could continue the maneuvers indefinitely (assuming unlimited fuel and battery endurance). The completed maneuvers are significantly more challenging than previously completed maneuvers.

In the (forward) *flip*, the helicopter rotates 360 degrees forward around its lateral axis (the axis going from the right to the left of the helicopter). To prevent altitude loss during the maneuver, the helicopter pushes itself back up by using the (inverted) main rotor thrust halfway through the flip. In the (right) axial *roll* the helicopter rotates 360 degrees around its longitudinal axis (the axis going from the back to the front of the helicopter). Similarly to the flip, the helicopter prevents altitude

loss by pushing itself back up by using the (inverted) main rotor thrust halfway through the roll. In the *tail-in funnel*, the helicopter repeatedly flies a circle sideways with the tail pointing to the center of the circle. For the trajectory to be a funnel maneuver, the helicopter speed and the circle radius are chosen such that the helicopter must pitch up steeply to stay in the circle. The *nose-in funnel* is similar to the tail-in funnel, the difference being that the nose points to the center of the circle throughout the maneuver.

The remainder of this paper is organized as follows: Section 2 explains how we learn a model from flight data. The section considers both the problem of data collection, for which we use an apprenticeship learning approach, as well as the problem of estimating the model from data. Section 3 explains our control design. We explain differential dynamic programming as applied to our helicopter. We discuss our apprenticeship learning approach to choosing the reward function, as well as other design decisions and lessons learned. Section 4 describes our helicopter platform and our experimental results. Section 5 concludes the paper. Movies of our autonomous helicopter flights are available at the following webpage:

<div align="center">

`http://www.cs.stanford.edu/~pabbeel/heli-nips2006`.

</div>

## 2 Learning a Helicopter Model from Flight Data

### 2.1 Data Collection

The $E^3$-family of algorithms [12] and its extensions [11, 7, 10] are the state of the art RL algorithms for autonomous data collection. They proceed by generating "exploration" policies, which try to visit inaccurately modeled parts of the state space. Unfortunately, such exploration policies do not even try to fly the helicopter well, and thus would invariably lead to crashes. Thus, instead, we use the apprenticeship learning algorithm proposed in [3], which proceeds as follows:

1. Collect data from a human pilot flying the desired maneuvers with the helicopter. Learn a model from the data.
2. Find a controller that works in simulation based on the current model.
3. Test the controller on the helicopter. If it works, we are done. Otherwise, use the data from the test flight to learn a new (improved) model and go back to Step 2.

This procedure has similarities with model-based RL and with the common approach in control to first perform system identification and then find a controller using the resulting model. However, the key insight from [3] is that this procedure is guaranteed to converge to expert performance in a polynomial number of iterations. In practice we have needed at most three iterations. Importantly, unlike the $E^3$ family of algorithms, this procedure never uses explicit exploration policies. We only have to test controllers that try to fly as well as possible (according to the current simulator).

### 2.2 Model Learning

The helicopter state $s$ comprises its position $(x, y, z)$, orientation (expressed as a unit quaternion), velocity $(\dot{x}, \dot{y}, \dot{z})$ and angular velocity $(\omega_x, \omega_y, \omega_z)$. The helicopter is controlled by a 4-dimensional action space $(u_1, u_2, u_3, u_4)$. By using the cyclic pitch $(u_1, u_2)$ and tail rotor $(u_3)$ controls, the pilot can rotate the helicopter around each of its main axes and bring the helicopter to any orientation. This allows the pilot to direct the thrust of the main rotor in any particular direction (and thus fly in any particular direction). By adjusting the collective pitch angle (control input $u_4$), the pilot can adjust the thrust generated by the main rotor. For a positive collective pitch angle the main rotor will blow air downward relative to the helicopter. For a negative collective pitch angle the main rotor will blow air upward relative to the helicopter. The latter allows for inverted flight.

Following [1] we learn a model from flight data that predicts accelerations as a function of the current state and inputs. Accelerations are then integrated to obtain the helicopter states over time. The key idea from [1] is that, after subtracting out the effects of gravity, the forces and moments acting on the helicopter are independent of position and orientation of the helicopter, when expressed in a "body coordinate frame", a coordinate frame attached to the body of the helicopter. This observation allows us to significantly reduce the dimensionality of the model learning problem. In particular, we use the following model:

$$
\begin{aligned}
\ddot{x}^b &= A_x \dot{x}^b + g_x^b + w_x, \\
\ddot{y}^b &= A_y \dot{y}^b + g_y^b + D_0 + w_y, \\
\ddot{z}^b &= A_z \dot{z}^b + g_z^b + C_4 u_4 + E_0 \|(\dot{x}^b, \dot{y}^b, \dot{z}^b)\|_2 + D_4 + w_z,
\end{aligned}
$$

$$\dot{\omega}_x^b = B_x \omega_x^b + C_1 u_1 + D_1 + w_{\omega_x},$$
$$\dot{\omega}_y^b = B_y \omega_y^b + C_2 u_2 + C_{24} u_4 + D_2 + w_{\omega_y},$$
$$\dot{\omega}_z^b = B_z \omega_z^b + C_3 u_3 + C_{34} u_4 + D_3 + w_{\omega_z}.$$

By our convention, the superscripts $b$ indicate that we are using a body coordinate frame with the x-axis pointing forwards, the y-axis pointing to the right and the z-axis pointing down with respect to the helicopter. We note our model explicitly encodes the dependence on the gravity vector $(g_x^b, g_y^b, g_z^b)$ and has a sparse dependence of the accelerations on the current velocities, angular rates and inputs. This sparse dependence was obtained by scoring different models by their simulation accuracy over time intervals of two seconds (similar to [4]). We estimate the coefficients $A_., B_., C_., D_.$ and $E_.$ from helicopter flight data. First we obtain state and acceleration estimates using a highly optimized extended Kalman filter, then we use linear regression to estimate the coefficients. The terms $w_x, w_y, w_z, w_{\omega_x}, w_{\omega_y}, w_{\omega_z}$ are zero mean Gaussian random variables, which represent the perturbations to the accelerations due to noise (or unmodeled effects). Their variances are estimated as the average squared prediction error on the flight data we collected.

The coefficient $D_0$ captures sideways acceleration of the helicopter due to thrust generated by the tail rotor. The term $E_0 \|(\dot{x}^b, \dot{y}^b, \dot{z}^b)\|_2$ models translational lift: the additional lift the helicopter gets when flying at higher speed. Specifically, during hover, the helicopter's rotor imparts a downward velocity on the air above and below it. This downward velocity reduces the effective pitch (angle of attack) of the rotor blades, causing less lift to be produced [14, 20]. As the helicopter transitions into faster flight, this region of altered airflow is left behind and the blades enter "clean" air. Thus, the angle of attack is higher and more lift is produced for a given choice of the collective control ($u_4$). The translational lift term was important for modeling the helicopter dynamics during the funnels. The coefficient $C_{24}$ captures the pitch acceleration due to main rotor thrust. This coefficient is non-zero since (after equipping our helicopter with our sensor packages) the center of gravity is further backward than the center of main rotor thrust.

There are two notable differences between our model and the most common previously proposed models (e.g., [15, 8]): (1) Our model does not include the inertial coupling between different axes of rotation. (2) Our model's state does not include the blade-flapping angles, which are the angles the rotor blades make with the helicopter body while sweeping through the air. Both inertial coupling and blade flapping have previously been shown to improve accuracy of helicopter models for other RC helicopters. However, extensive attempts to incorporate them into our model have not led to improved simulation accuracy. We believe the effects of inertial coupling to be very limited since the flight regimes considered do not include fast rotation around more than one main axis simultaneously. We believe that—at the 0.1s time scale used for control—the blade flapping angles' effects are sufficiently well captured by using a first order model from cyclic inputs to roll and pitch rates. Such a first order model maps cyclic inputs to angular accelerations (rather than the steady state angular rate), effectively capturing the delay introduced by the blades reacting (moving) first before the helicopter body follows.

## 3 Controller Design

### 3.1 Reinforcement Learning Formalism and Differential Dynamic Programming (DDP)

A reinforcement learning problem (or optimal control problem) can be described by a Markov decision process (MDP), which comprises a sextuple $(S, \mathcal{A}, T, H, s(0), R)$. Here $S$ is the set of states; $\mathcal{A}$ is the set of actions or inputs; $T$ is the dynamics model, which is a set of probability distributions $\{P_{su}^t\}$ ($P_{su}^t(s'|s, u)$ is the probability of being in state $s'$ at time $t+1$ given the state and action at time $t$ are $s$ and $u$); $H$ is the horizon or number of time steps of interest; $s(0) \in S$ is the initial state; $R : S \times \mathcal{A} \to \mathbb{R}$ is the reward function.

A policy $\pi = (\mu_0, \mu_1, \cdots, \mu_H)$ is a tuple of mappings from the set of states $S$ to the set of actions $\mathcal{A}$, one mapping for each time $t = 0, \cdots, H$. The expected sum of rewards when acting according to a policy $\pi$ is given by: $\mathrm{E}[\sum_{t=0}^{H} R(s(t), u(t))|\pi]$. The optimal policy $\pi^*$ for an MDP $(S, \mathcal{A}, T, H, s(0), R)$ is the policy that maximizes the expected sum of rewards. In particular, the optimal policy is given by $\pi^* = \arg\max_\pi \mathrm{E}[\sum_{t=0}^{H} R(s(t), u(t))|\pi]$.

The linear quadratic regulator (LQR) control problem is a special class of MDPs, for which the optimal policy can be computed efficiently. In LQR the set of states is given by $S = \mathbb{R}^n$, the set of actions/inputs is given by $\mathcal{A} = \mathbb{R}^p$, and the dynamics model is given by:
$$s(t+1) = A(t)s(t) + B(t)u(t) + w(t),$$

where for all $t = 0, \ldots, H$ we have that $A(t) \in \mathbb{R}^{n \times n}, B(t) \in \mathbb{R}^{n \times p}$ and $w(t)$ is a zero mean random variable (with finite variance). The reward for being in state $s(t)$ and taking action/input $u(t)$ is given by:

$$-s(t)^\top Q(t) s(t) - u(t)^\top R(t) u(t).$$

Here $Q(t), R(t)$ are positive semi-definite matrices which parameterize the reward function. It is well-known that the optimal policy for the LQR control problem is a linear feedback controller which can be efficiently computed using dynamic programming. Although the standard formulation presented above assumes the all-zeros state is the most desirable state, the formalism is easily extended to the task of tracking a desired trajectory $s_0^*, \ldots, s_H^*$. The standard extension (which we use) expresses the dynamics and reward function as a function of the error state $e(t) = s(t) - s^*(t)$ rather than the actual state $s(t)$. (See, e.g., [5], for more details on linear quadratic methods.)

Differential dynamic programming (DDP) approximately solves general continuous state-space MDPs by iterating the following two steps:

1. Compute a linear approximation to the dynamics and a quadratic approximation to the reward function around the trajectory obtained when using the current policy.
2. Compute the optimal policy for the LQR problem obtained in Step 1 and set the current policy equal to the optimal policy for the LQR problem.

In our experiments, we have a quadratic reward function, thus the only approximation made in the first step is the linearization of the dynamics. To bootstrap the process, we linearized around the target trajectory in the first iteration.[1]

### 3.2 DDP Design Choices

**Error state.** We use the following error state $e = (\dot{x}^b - (\dot{x}^b)^*, \dot{y}^b - (\dot{y}^b)^*, \dot{z}^b - (\dot{z}^b)^*, x - x^*, y - y^*, z - z^*, \omega_x^b - (\omega_y^b)^*, \omega_y^b - (\omega_y^b)^*, \omega_z^b - (\omega_z^b)^*, \Delta_q)$. Here $\Delta_q$ is the axis-angle representation of the rotation that transforms the coordinate frame of the target orientation into the coordinate frame of the actual state. This axis angle representation results in the linearizations being more accurate approximations of the non-linear model since the axis angle representation maps more directly to the angular rates than naively differencing the quaternions or Euler angles.

**Cost for change in inputs.** Using DDP as thus far explained resulted in unstable controllers on the real helicopter: The controllers tended to rapidly switch between low and high values, which resulted in poor flight performance. Similar to frequency shaping for LQR controllers (see, e.g., [5]), we added a term to the reward function that penalizes the change in inputs over consecutive time steps.

**Controller design in two phases.** Adding the cost term for the change in inputs worked well for the funnels. However flips and rolls do require some fast changes in inputs. To still allow aggressive maneuvering, we split our controller design into two phases. In the first phase, we used DDP to find the open-loop input sequence that would be optimal in the noise-free setting. (This can be seen as a planning phase and is similar to designing a feedforward controller in classical control.) In the second phase, we used DDP to design our actual flight controller, but we now redefine the inputs as the deviation from the nominal open-loop input sequence. Penalizing for changes in the new inputs penalizes only unplanned changes in the control inputs.

**Integral control.** Due to modeling error and wind, the controllers (so far described) have non-zero steady-state error. Each controller generated by DDP is designed using linearized dynamics. The orientation used for linearization greatly affects the resulting linear model. As a consequence, the linear model becomes significantly worse an approximation with increasing orientation error. This in turn results in the control inputs being less suited for the current state, which in turn results in larger orientation error, etc. To reduce the steady-state orientation errors—similar to the I term

in PID control—we augment the state vector with integral terms for the orientation errors. More specifically, the state vector at time $t$ is augmented with $\sum_{\tau=0}^{t-1} 0.99^{t-\tau} \Delta_q(\tau)$. Our funnel controllers performed significantly better with integral control. For the flips and rolls the integral control seemed to matter less.[2]

**Factors affecting control performance.** Our simulator included process noise (Gaussian noise on the accelerations as estimated when learning the model from data), measurement noise (Gaussian noise on the measurements as estimated from the Kalman filter residuals), as well as the Kalman filter and the low-pass filter, which is designed to remove the high-frequency noise from the IMU measurements.[3] Simulator tests showed that the low-pass filter's latency and the noise in the state estimates affect the performance of our controllers most. Process noise on the other hand did not seem to affect performance very much.

### 3.3 Trade-offs in the reward function

Our reward function contained 24 features, consisting of the squared error state variables, the squared inputs, the squared change in inputs between consecutive timesteps, and the squared integral of the error state variables. For the reinforcement learning algorithm to find a controller that flies "well," it is critical that the correct trade-off between these features is specified. To find the correct trade-off between the 24 features, we first recorded a pilot's flight. Then we used the apprenticeship learning via inverse reinforcement learning algorithm [2]. The inverse RL algorithm iteratively provides us with reward weights that result in policies that bring us closer to the expert. Unfortunately the reward weights generated throughout the iterations of the algorithm are often unsafe to fly on the helicopter. Thus rather than strictly following the inverse RL algorithm, we hand-chose reward weights that (iteratively) bring us closer to the expert human pilot by increasing/decreasing the weights for those features that stood out as mostly different from the expert (following the philosophy, but not the strict formulation of the inverse RL algorithm). The algorithm still converged in a small number of iterations.

## 4 Experiments

*Videos of all of our maneuvers are available at the URL provided in the introduction.*

### 4.1 Experimental Platform

The helicopter used is an XCell Tempest, a competition-class aerobatic helicopter (length 54", height 19", weight 13 lbs), powered by a 0.91-size, two-stroke engine. Figure 2 (c) shows a close-up of the helicopter. We instrumented the helicopter with a Microstrain 3DM-GX1 orientation sensor, and a Novatel RT2 GPS receiver. The Microstrain package contains triaxial accelerometers, rate gyros, and magnetometers. The Novatel RT2 GPS receiver uses carrier-phase differential GPS to provide real-time position estimates with approximately 2cm accuracy *as long as its antenna is pointing at the sky*. To maintain position estimates throughout the flips and rolls, we have used two different setups. Originally, we used a purpose-built cluster of four U-Blox LEA-4T GPS receivers/antennas for velocity sensing. The system provides velocity estimates with standard deviation of approximately 1 cm/sec (when stationary) and 10cm/sec (during our aerobatic maneuvers). Later, we used three PointGrey DragonFly2 cameras that track the helicopter from the ground. This setup gives us 25cm accurate position measurements. For extrinsic camera calibration we collect data from the Novatel RT2 GPS receiver while in view of the cameras. A computer on the ground uses a Kalman filter to estimate the state from the sensor readings. Our controllers generate control commands at 10Hz.

### 4.2 Experimental Results

For each of the maneuvers, the initial model is learned by collecting data from a human pilot flying the helicopter. Our sensing setup is significantly less accurate when flying upside-down, so all data for model learning is collected from upright flight. The model used to design the flip and roll controllers is estimated from 5 minutes of flight data during which the pilot performs frequency sweeps on each of the four control inputs (which covers as similar a flight regime as possible without having to invert the helicopter). For the funnel controllers, we learn a model from the same frequency sweeps and from our pilot flying the funnels. For the rolls and flips the initial model was sufficiently accurate for control. For the funnels, our initial controllers did not perform as well, and we performed two iterations of the apprenticeship learning algorithm described in Section 2.1.

### 4.2.1 Flip

In the ideal forward flip, the helicopter rotates 360 degrees forward around its lateral axis (the axis going from the right to the left of the helicopter) while staying in place. The top row of Figure 1 (a) shows a series of snapshots of our helicopter during an autonomous flip. In the first frame, the helicopter is hovering upright autonomously. Subsequently, it pitches forward, eventually becoming vertical. At this point, the helicopter does not have the ability to counter its descent since it can only produce thrust in the direction of the main rotor. The flip continues until the helicopter is completely inverted. At this moment, the controller must apply negative collective to regain altitude lost during the half-flip, while continuing the flip and returning to the upright position.

We chose the entries of the cost matrices $Q$ and $R$ by hand, spending about an hour to get a controller that could flip indefinitely in our simulator. The initial controller oscillated in reality whereas our human piloted flips do not have any oscillation, so (in accordance with the inverse RL procedure, see Section 3.3) we increased the penalty for changes in inputs over consecutive time steps, resulting in our final controller.

### 4.2.2 Roll

In the ideal axial roll, the helicopter rotates 360 degrees around its longitudinal axis (the axis going from the back to the front of the helicopter) while staying in place. The bottom row of Figure 1 (b) shows a series of snapshots of our helicopter during an autonomous roll. In the first frame, the helicopter is hovering upright autonomously. Subsequently it rolls to the right, eventually becoming inverted. When inverted, the helicopter applies negative collective to regain altitude lost during the first half of the roll, while continuing the roll and returning to the upright position. We used the same cost matrices as for the flips.

### 4.2.3 Tail-In Funnel

The tail-in funnel maneuver is essentially a medium to high speed circle flown sideways, with the tail of the helicopter pointed towards the center of the circle. Throughout, the helicopter is pitched backwards such that the main rotor thrust not only compensates for gravity, but also provides the centripetal acceleration to stay in the circle. For a funnel of radius $r$ at velocity $v$ the centripetal acceleration is $v^2/r$, so—assuming the main rotor thrust only provides the centripetal acceleration and compensation for gravity—we obtain a pitch angle $\theta = \mathrm{atan}(v^2/(rg))$. The maneuver is named after the path followed by the length of the helicopter, which sweeps out a surface similar to that of an inverted cone (or funnel). [4] For the funnel reported in this paper, we had $H = 80$ s, $r = 5$ m, and $v = 5.3$ m/s (which yields a 30 degree pitch angle during the funnel). Figure 1 (c) shows an overlay of snapshots of the helicopter throughout a tail-in funnel.

The defining characteristic of the funnel is repeatability—the ability to pass consistently through the same points in space after multiple circuits. Our autonomous funnels are significantly more accurate than funnels flown by expert human pilots. Figure 2 (a) shows a complete trajectory in (North, East) coordinates. In figure 2 (b) we superimposed the heading of the helicopter on a partial trajectory (showing the entire trajectory with heading superimposed gives a cluttered plot). Our autonomous funnels have an RMS position error of 1.5m and an RMS heading error of 15 degrees throughout the twelve circuits flown. Expert human pilots can maintain this performance at most through one or two circuits. [5]

### 4.2.4 Nose-In Funnel

The nose-in funnel maneuver is very similar to the tail-in funnel maneuver, except that the nose points to the center of the circle, rather than the tail. Our autonomous nose-in funnel controller results in highly repeatable trajectories (similar to the tail-in funnel), and it achieves a level of performance that is difficult for a human pilot to match. Figure 1 (d) shows an overlay of snapshots throughout a nose-in funnel.

## 5    Conclusion

To summarize, we presented our successful DDP-based control design for four new aerobatic maneuvers: forward flip, sideways roll (at low speed), tail-in funnel, and nose-in funnel. The key design decisions for the DDP-based controller to fly our helicopter successfully are the following:

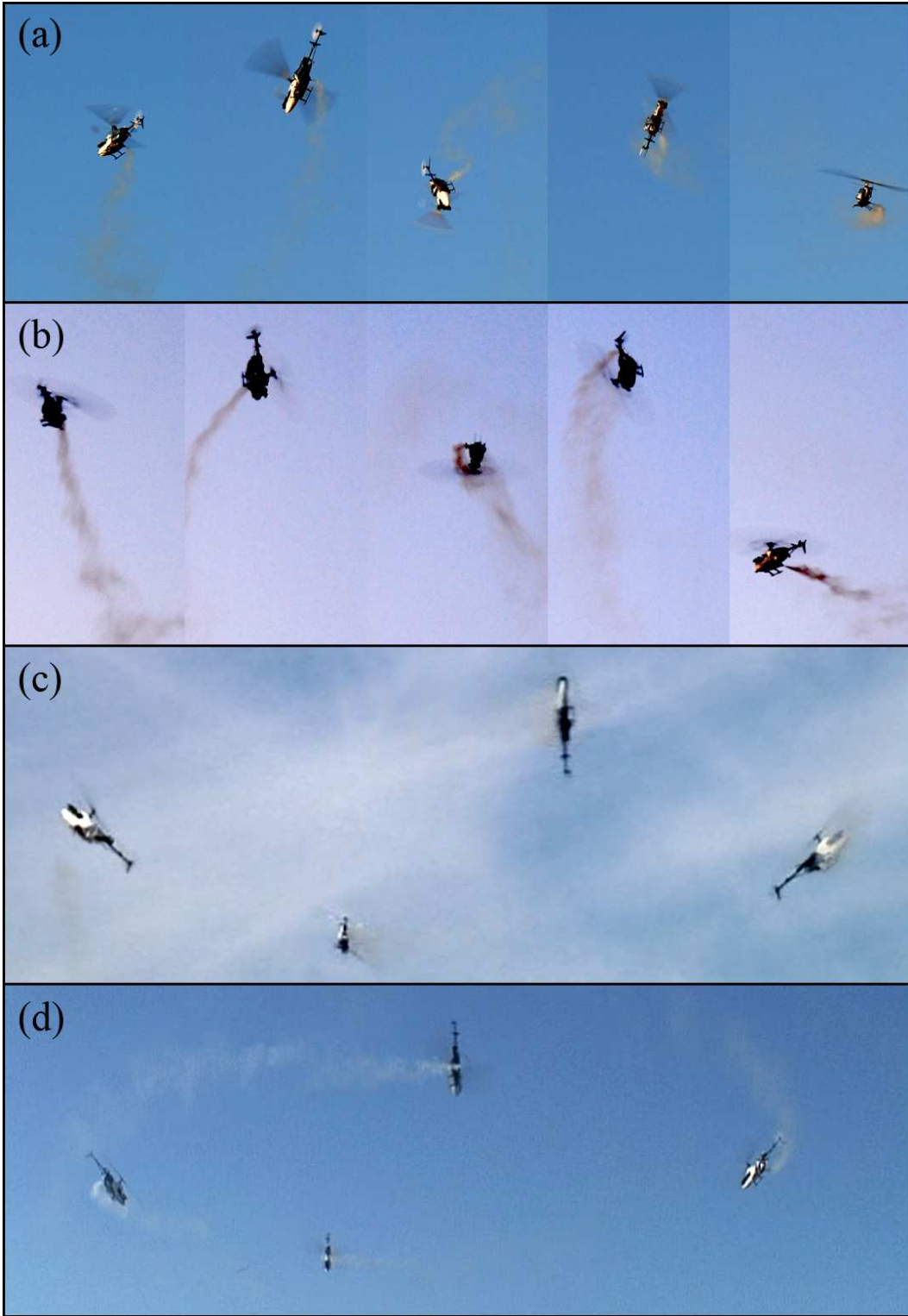

Figure 1: (Best viewed in color.) (a) Series of snapshots throughout an autonomous flip. (b) Series of snapshots throughout an autonomous roll. (c) Overlay of snapshots of the helicopter throughout a tail-in funnel. (d) Overlay of snapshots of the helicopter throughout a nose-in funnel. (See text for details.)

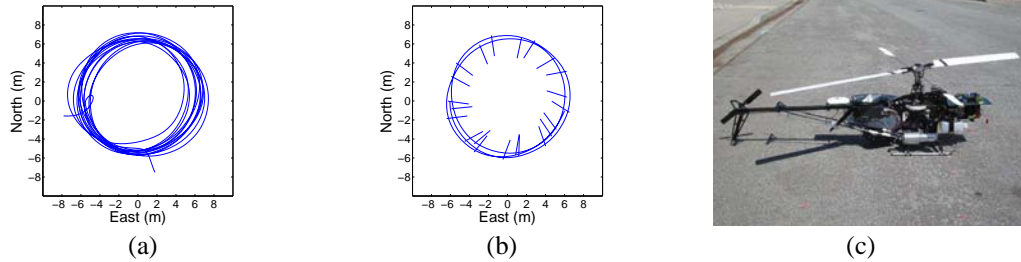

Figure 2: (a) Trajectory followed by the helicopter during tail-in funnel. (b) Partial tail-in funnel trajectory with heading marked. (c) Close-up of our helicopter. (See text for details.)

We penalized for rapid changes in actions/inputs over consecutive time steps. We used apprenticeship learning algorithms, which take advantage of an expert demonstration, to determine the reward function and to learn the model. We used a two-phase control design: the first phase plans a feasible trajectory, the second phase designs the actual controller. Integral penalty terms were included to reduce steady-state error. To the best of our knowledge, these are the most challenging autonomous flight maneuvers achieved to date.

## Acknowledgments

We thank Ben Tse for piloting our helicopter and working on the electronics of our helicopter. We thank Mark Woodward for helping us with the vision system.

## Footnotes

[1]For the flips and rolls this simple initialization did not work: Due to the target trajectory being too far from feasible, the control policy obtained in the first iteration of DDP ended up following a trajectory for which the linearization is inaccurate. As a consequence, the first iteration's control policy (designed for the time-varying linearized models along the target trajectory) was unstable in the non-linear model and DDP failed to converge. To get DDP to converge to good policies we slowly changed the model from a model in which control is trivial to the actual model. In particular, we change the model such that the next state is $\alpha$ times the target state plus $1 - \alpha$ times the next state according to the true model. By slowly varying $\alpha$ from 0.999 to zero throughout DDP iterations, the linearizations obtained throughout are good approximations and DDP converges to a good policy.

[2]When adding the integrated error in position to the cost we did not experience any benefits. Even worse, when increasing its weight in the cost function, the resulting controllers were often unstable.

[3]The high frequency noise on the IMU measurements is caused by the vibration of the helicopter. This vibration is mostly caused by the blades spinning at 25Hz.

[4]The maneuver is actually broken into three parts: an accelerating leg, the funnel leg, and a decelerating leg. During the accelerating and decelerating legs, the helicopter accelerates at $a_{\max} (= 0.8 m/s^2)$ along the circle.

[5]Without the integral of heading error in the cost function we observed significantly larger heading errors of 20-40 degrees, which resulted in the linearization being so inaccurate that controllers often failed entirely.

## References

[1] P. Abbeel, Varun Ganapathi, and Andrew Y. Ng. Learning vehicular dynamics with application to modeling helicopters. In *NIPS 18*, 2006.

[2] P. Abbeel and A. Y. Ng. Apprenticeship learning via inverse reinforcement learning. In *Proc. ICML*, 2004.

[3] P. Abbeel and A. Y. Ng. Exploration and apprenticeship learning in reinforcement learning. In *Proc. ICML*, 2005.

[4] P. Abbeel and A. Y. Ng. Learning first order Markov models for control. In *NIPS 18*, 2005.

[5] B. Anderson and J. Moore. *Optimal Control: Linear Quadratic Methods*. Prentice-Hall, 1989.

[6] J. Bagnell and J. Schneider. Autonomous helicopter control using reinforcement learning policy search methods. In *International Conference on Robotics and Automation*. IEEE, 2001.

[7] Ronen I. Brafman and Moshe Tennenholtz. R-max, a general polynomial time algorithm for near-optimal reinforcement learning. *Journal of Machine Learning Research*, 2002.

[8] V. Gavrilets, I. Martinos, B. Mettler, and E. Feron. Flight test and simulation results for an autonomous aerobatic helicopter. In *AIAA/IEEE Digital Avionics Systems Conference*, 2002.

[9] V. Gavrilets, B. Mettler, and E. Feron. Human-inspired control logic for automated maneuvering of miniature helicopter. *Journal of Guidance, Control, and Dynamics*, 27(5):752–759, 2004.

[10] S. Kakade, M. Kearns, and J. Langford. Exploration in metric state spaces. In *Proc. ICML*, 2003.

[11] M. Kearns and D. Koller. Efficient reinforcement learning in factored MDPs. In *Proc. IJCAI*, 1999.

[12] M. Kearns and S. Singh. Near-optimal reinforcement learning in polynomial time. *Machine Learning Journal*, 2002.

[13] M. La Civita, G. Papageorgiou, W. C. Messner, and T. Kanade. Design and flight testing of a high-bandwidth $\mathcal{H}_\infty$ loop shaping controller for a robotic helicopter. *Journal of Guidance, Control, and Dynamics*, 29(2):485–494, March-April 2006.

[14] J. Leishman. *Principles of Helicopter Aerodynamics*. Cambridge University Press, 2000.

[15] B. Mettler, M. Tischler, and T. Kanade. System identification of small-size unmanned helicopter dynamics. In *American Helicopter Society, 55th Forum*, 1999.

[16] A. Y. Ng, A. Coates, M. Diel, V. Ganapathi, J. Schulte, B. Tse, E. Berger, and E. Liang. Autonomous inverted helicopter flight via reinforcement learning. In *Int'l Symposium on Experimental Robotics*, 2004.

[17] Andrew Y. Ng, H. Jin Kim, Michael Jordan, and Shankar Sastry. Autonomous helicopter flight via reinforcement learning. In *NIPS 16*, 2004.

[18] Jonathan M. Roberts, Peter I. Corke, and Gregg Buskey. Low-cost flight control system for a small autonomous helicopter. In *IEEE Int'l Conf. on Robotics and Automation*, 2003.

[19] S. Saripalli, J. F. Montgomery, and G. S. Sukhatme. Visually-guided landing of an unmanned aerial vehicle. *IEEE Transactions on Robotics and Autonomous Systems*, 2003.

[20] J. Seddon. *Basic Helicopter Aerodynamics*. AIAA Education Series. America Institute of Aeronautics and Astronautics, 1990.
